# Divisive Normalization, Line Attractor Networks and Ideal Observers

**Sophie Deneve[1] Alexandre Pouget[1], and P.E. Latham[2]**
[1]Georgetown Institute for Computational and Cognitive Sciences,
Georgetown University, Washington, DC 20007-2197
[2]Dpt of Neurobiology, UCLA, Los Angeles, CA 90095-1763, U.S.A.

## Abstract

Gain control by divisive inhibition, a.k.a. divisive normalization, has been proposed to be a general mechanism throughout the visual cortex. We explore in this study the statistical properties of this normalization in the presence of noise. Using simulations, we show that divisive normalization is a close approximation to a maximum likelihood estimator, which, in the context of population coding, is the same as an ideal observer. We also demonstrate analytically that this is a general property of a large class of nonlinear recurrent networks with line attractors. Our work suggests that divisive normalization plays a critical role in noise filtering, and that every cortical layer may be an ideal observer of the activity in the preceding layer.

Information processing in the cortex is often formalized as a sequence of a linear stages followed by a nonlinearity. In the visual cortex, the nonlinearity is best described by squaring combined with a divisive pooling of local activities. The divisive part of the nonlinearity has been extensively studied by Heeger and colleagues [1], and several authors have explored the role of this normalization in the computation of high order visual features such as orientation of edges or first and second order motion[4]. We show in this paper that divisive normalization can also play a role in noise filtering. More specifically, we demonstrate through simulations that networks implementing this normalization come close to performing maximum likelihood estimation. We then demonstrate analytically that the ability to perform maximum likelihood estimation, and thus efficiently extract information from a population of noisy neurons, is a property exhibited by a large class of networks.

Maximum likelihood estimation is a framework commonly used in the theory of ideal observers. A recent example comes from the work of Itti et al., 1998, who have shown that it is possible to account for the behavior of human subjects in simple discrimination tasks. Their model comprised two *distinct* stages: 1) a network

which models the noisy response of neurons with tuning curves to orientation and spatial frequency combined with divisive normalization, and 2) an ideal observer (a maximum likelihood estimator) to read out the population activity of the network.

Our work suggests that there is no need to distinguish between these two stages, since, as we will show, divisive normalization comes close to providing a maximum likelihood estimation. More generally, we propose that there may not be any part of the cortex that acts as an ideal observer for patterns of activity in sensory areas but, instead, that each cortical layer acts as an ideal observer of the activity in the preceding layer.

# 1 The network

Our network is a simplified model of a cortical hypercolumn for spatial frequency and orientation. It consists of a two dimensional array of units in which each unit is indexed by its preferred orientation, $\theta_i$, and spatial frequency, $\lambda_j$.

## 1.1 LGN model

Units in the cortical layer are assumed to receive direct inputs from the lateral geniculate nucleus (LGN). Here we do not model explicitly the LGN, but focus instead on the pooled LGN input onto each cortical unit. The input to each unit is denoted $a_{ij}$. We distinguish between the *mean* pooled LGN input, $f_{ij}(\theta, \lambda)$, as a function of orientation, $\theta$, and spatial frequency, $\lambda$, and the *noise* distribution around this mean, $P(a_{ij}|\theta, \lambda)$.

In response to a stimulus of orientation, $\theta$, spatial frequency, $\lambda$, and contrast, $C$, the mean LGN input onto unit $ij$ is a circular Gaussian with a small amount of spontaneous activity, $\nu$:

$$f_{ij}(\theta, \lambda) = KC \exp\left(\frac{\cos(\lambda - \lambda_j) - 1}{\sigma_\lambda^2} + \frac{\cos(\theta - \theta_i) - 1}{\sigma_\theta^2}\right) + \nu, \qquad (1)$$

where K is a constant. Note that spatial frequency is treated as a periodic variable; this was done for convenience only and should have negligible effects on our results as long as we keep $\lambda$ far from $2\pi n$, $n$ an integer.

On any given trial the LGN input to cortical unit $ij$, $a_{ij}$, is sampled from a Gaussian noise distribution with variance $\sigma_{ij}^2$:

$$p(a_{ij}|\theta, \lambda) = \frac{1}{\sqrt{2\pi\sigma_{ij}^2}} \exp\left(-\frac{[a_{ij} - f_{ij}(\theta, \lambda)]^2}{2\sigma_{ij}^2}\right). \qquad (2)$$

In our simulations, the variance of the noise was either kept fixed ($\sigma_{ij}^2 = \sigma^2$) or set to the mean activity ($\sigma_{ij}^2 = f_{ij}(\theta, \lambda)$). The latter is more consistent with the noise that has been measured experimentally in the cortex. We show in figure 1-A an example of a noisy LGN pattern of activity.

## 1.2 Cortical Model: Divisive Normalization

Activities in the cortical layer are updated over time according to:

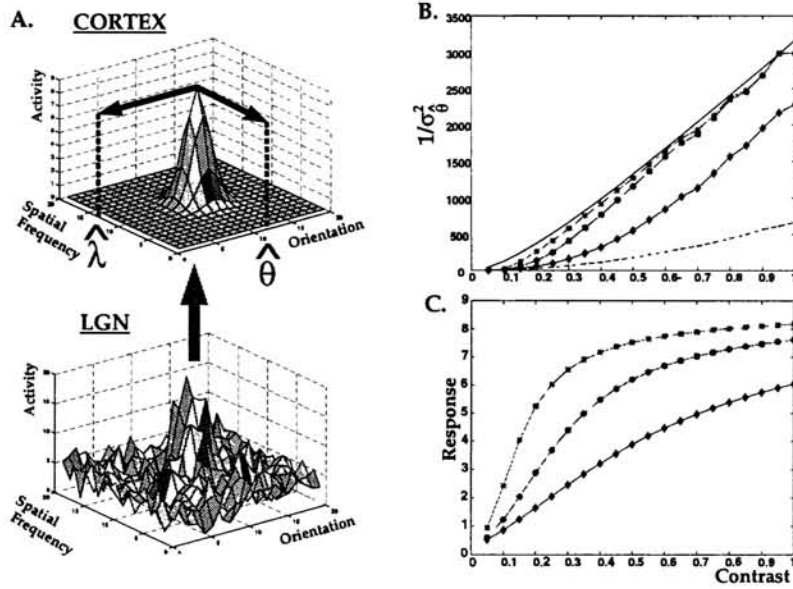

Figure 1: A- LGN input (bottom) and stable hill in the cortical network after relaxation (top). The position of the stable hill can be used to estimate orientation ($\hat{\theta}$) and spatial frequency ($\hat{\lambda}$). B- Inverse of the variance of the network estimate for orientation using Gaussian noise with variance equal to the mean as a function of contrast and number of iterations (0, dashed; 1, diamond; 2, circle; and 3, square). The continuous curve corresponds to the theoretical upper bound on the inverse of the variance (i.e. an ideal observer). C- Gain curve for contrast for the cortical units after 1, 2 and 3 iterations.

$$u_{ij}(t+1) = \sum_{kl} w_{ij,kl} o_{kl}(t), \quad o_{ij}(t+1) = \frac{u_{ij}(t+1)^2}{S + \mu \sum_{kl} u_{kl}(t+1)^2}, \qquad (3)$$

where $\{w_{ij,kl}\}$ are the *filtering* weights, $o_{ij}(t)$ is the activity of unit $ij$ at time $t$, $S$ is a constant, and $\mu$ is what we call the *divisive inhibition* weight. The filtering weights implement a two dimensional Gaussian filter:

$$w_{ij,kl} = w_{i-k,j-l} = K_w \exp\left( \frac{\cos[2\pi(i-k)/P] - 1}{\sigma_{w\theta}^2} + \frac{\cos[2\pi(j-l)/P] - 1}{\sigma_{w\lambda}^2} \right) \quad (4)$$

where $K_w$ is a constant, $\sigma_{w\theta}$ and $\sigma_{w\lambda}$ control the width of the filtering weights, and there are $P^2$ units.

On each iteration the activity is filtered by the weights, squared, and then normalized by the total local activity. Divisive normalization *per se* only involves the squaring and division by local activity. We have added the filtering weights to obtain a local pooling of activity between cells with similar preferred orientations and spatial frequencies. This pooling can easily be implemented with cortical lateral connections and it is reasonable to think that such a pooling takes place in the cortex.

## 2   Simulation Results

Our simulations consist of iterating equation 3 with initial conditions determined by the presentation orientation and spatial frequency. The initial conditions are chosen as follows: For a given presentation angle, $\theta_0$, and spatial frequency, $\lambda_0$, determine the mean cortical activity, $f_{ij}(\theta_0, \lambda_0)$, via equation 1. Then generate the actual cortical activity, $\{a_{ij}\}$, by sampling from the distribution given in equation 2. This serves as our set of initial conditions: $o_{ij}(t=0) = a_{ij}$.

Iterating equation 3 with the above initial conditions, we found that for very low contrast the activity of all cortical units decayed to zero. Above some contrast threshold, however, the activities converged to a smooth stable hill (see figure 1-A for an example with parameters $\sigma_{w\theta} = \sigma_{w\lambda} = \sigma_\theta = \sigma_\lambda = 1/\sqrt{8}$, $K = 74$, $C = 1$, $\mu = 0.01$). The width of the hill is controlled by the width of the filtering weights. Its peak, on the other hand, depends on the orientation and spatial frequency of the LGN input, $\theta_0$ and $\lambda_0$. The peak can thus be used to estimate these quantities (see figure 1-A). To compute the position of the final hill, we used a population vector estimator [3] although any unbiased estimator would work as well. In all cases we looked at, the network produced an unbiased estimate of $\theta_0$ and $\lambda_0$.

In our simulations we adjusted $\sigma_{w\theta}$ and $\sigma_{w\lambda}$ so that the stable hill had the same profile as the mean LGN input (equation 1). As a result, the tuning curves of the cortical units match the tuning curves specified by the pooled LGN input. For this case, we found that the estimate obtained from the network has a variance close to the theoretical minimum, known as the Cramér-Rao bound [3]. For Gaussian noise of fixed variance, the variance of the estimate was 16.6% above this bound, compared to 3833% for the population vector applied directly to the LGN input. In a 1D network (orientation alone), these numbers go to 12.9% for the network versus 613% for population vector. For Gaussian noise with variance proportional to the mean, the network was 8.8% above the bound, compared to 722% for the population vector applied directly to the input. These numbers are respectively 9% and 108% for the 1-D network. The network is therefore a close approximation to a maximum likelihood estimator, i.e., it is close to being an ideal observer of the LGN activity with respect to orientation and spatial frequency.

As long as the contrast, $C$, was superthreshold, large variations in contrast did not affect our results (figure 1-B). However, the tuning of the network units to contrast after reaching the stable state was found to follow a step function whereas, for real neurons, the curves are better described by a sigmoid [2]. Improved agreement with experiment was achieved by taking only 2-3 iterations, at which point the performance of the network is close to optimal (figure 1-B) and the tuning curves to contrast are more realistic and closer to sigmoids (figure 1-C). Therefore, reaching a stable state is not required for optimal performance, and in fact leads to contrast tuning curves that are inconsistent with experiment.

## 3   Mathematical Analysis

We first prove that line attractor networks with sufficiently small noise are close approximations to a maximum likelihood estimator. We then show how this result applies to our simulations with divisive normalization.

### 3.1 General Case: Line Attractor Networks

Let $\mathbf{o}_n$ be the activity vector (denoted by bold type) at discrete time, $n$, for a set of $P$ interconnected units. We consider a one dimensional network, i.e., only one feature is encoded; generalization to multidimensional networks is straightforward. A generic mapping for this network may be written

$$\mathbf{o}_{n+1} = \mathbf{H}(\mathbf{o}_n) \tag{5}$$

where $\mathbf{H}$ is a nonlinear function. We assume that this mapping admits a line attractor, which we denote $\mathbf{G}(\theta)$, for which $\mathbf{G}(\theta) = \mathbf{H}(\mathbf{G}(\theta))$ where $\theta$ is a continuous variable.[1] Let the initial state of the network be a function of the presentation parameter, $\theta_0$, plus noise,

$$\mathbf{o}_0 = \mathbf{F}(\theta_0) + \mathbf{N} \tag{6}$$

where $\mathbf{F}(\theta_0)$ is the function used to generate the data (in our simulations this would correspond to the mean LGN input, equation 1). Iterating the mapping, equation 5, leads eventually to a point on the line attractor. Consequently, as $n \to \infty$, $\mathbf{o}_n \to \mathbf{G}(\hat{\theta})$. The parameter $\hat{\theta}$ provides an estimate of $\theta_0$.

To determine how well the network does we need to find $\delta\theta \equiv \hat{\theta} - \theta_0$ as a function of the noise, $\mathbf{N}$, then average over the noise to compute the mean and variance of $\delta\theta$. Because the mapping, equation 5, is nonlinear, this cannot be done exactly. For small noise, however, we can take a perturbative approach and expand around a point on the attractor. For line attractors there is no general method for choosing which point on the attractor to expand around. Our approach will be to expand around an arbitrary point, $\mathbf{G}(\theta)$, and choose $\theta$ by requiring that the quadratic terms be finite. Keeping terms up to quadratic order, equation 6 may be written

$$\mathbf{o}_n = \mathbf{G}(\theta) + \delta\mathbf{o}_n. \tag{7}$$

$$\delta\mathbf{o}_n = \mathbf{J}^n \cdot \delta\mathbf{o}_0 + \frac{1}{2}\sum_{m=0}^{n-1}(\mathbf{J}^m \cdot \delta\mathbf{o}_o) \cdot \mathbf{H}'' \cdot (\mathbf{J}^m \cdot \delta\mathbf{o}_o), \tag{8}$$

where $\mathbf{J}(\theta) \equiv [\partial_{\mathbf{G}(\theta)}\mathbf{H}(\mathbf{G}(\theta))]^T$ is the Jacobian (the subscript $T$ means transpose), $\mathbf{H}''$ is the Hessian of $\mathbf{H}$ evaluated at $\mathbf{G}(\theta)$ and a "$\cdot$" represents the standard dot product.

Because the mapping, equation 5, admits a line attractor, $\mathbf{J}$ has one eigenvalue equal to 1 and all others less than 1. Denote the eigenvector with eigenvalue 1 as $\mathbf{v}$ and its adjoint $\mathbf{v}^\dagger$: $\mathbf{J} \cdot \mathbf{v} = \mathbf{v}$ and $\mathbf{J}^T \cdot \mathbf{v}^\dagger = \mathbf{v}^\dagger$. It is not hard to show that $\mathbf{v} = \partial_\theta \mathbf{G}(\theta)$, up to a multiplicative constant. Since $\mathbf{J}$ has an eigenvalue equal to 1, to avoid the quadratic term in Eq. 8 approaching infinity as $n \to \infty$ we require that

$$\lim_{n \to \infty} \mathbf{J}^n \cdot \delta\mathbf{o}_0 = 0. \tag{9}$$

This equations has an important consequence: it implies that, to linear order, $\lim_{n\to\infty} \delta\mathbf{o}_n = 0$ (see equation 8), which in turn implies that $\mathbf{o}_\infty = \mathbf{G}(\theta)$ which, finally, implies that $\theta = \hat{\theta}$. Consequently we can find the network estimator of $\theta_0$, $\hat{\theta}$, by computing $\theta$. We now turn to that task.

It is straightforward to show that $\mathbf{J}^\infty = \mathbf{v}\mathbf{v}^\dagger$. Combining this expression for $\mathbf{J}$ with equation 9, using equation 7 to express $\delta\mathbf{o}_0$ in terms of $\mathbf{o}_0$ and $\mathbf{G}(\theta)$, and, finally using equation 6 to express $\mathbf{o}_0$ in terms of the initial mean activity, $\mathbf{F}(\theta_0)$, and the noise, $\mathbf{N}$, we find that

$$\mathbf{v}^\dagger(\theta) \cdot [\mathbf{F}(\theta_0) - \mathbf{G}(\theta) + \mathbf{N}] = 0. \tag{10}$$

Using $\theta_0 = \theta - \delta\theta$ and expanding $\mathbf{F}(\theta_0)$ to first order in $\delta\theta$ then yields

$$\delta\theta = \frac{\mathbf{v}^\dagger(\theta) \cdot [\mathbf{N} + \mathbf{F}(\theta) - \mathbf{G}(\theta)]}{\mathbf{v}^\dagger(\theta) \cdot \mathbf{F}'(\theta)}. \tag{11}$$

As long as $\mathbf{v}^\dagger$ is orthogonal to $\mathbf{F}(\theta) - \mathbf{G}(\theta)$, $\langle\delta\theta\rangle = 0$ and the estimator is unbiased. This must be checked on a case by case basis, but for the circularly symmetric networks we considered orthogonality is satisfied.

We can now calculate the variance of the network estimate, $\langle\delta\theta\rangle^2$. Assuming $\mathbf{v}^\dagger \cdot [\mathbf{F}(\theta) - \mathbf{G}(\theta)] = 0$, equation 11 implies that

$$\langle\delta\theta\rangle^2 = \frac{\mathbf{v}^\dagger \cdot \mathbf{R} \cdot \mathbf{v}^\dagger}{[\mathbf{v}^\dagger \cdot \mathbf{F}']^2}, \tag{12}$$

where a prime denotes a derivative with respect to $\theta$ and $\mathbf{R}$ is the covariance matrix of the noise, $\mathbf{R} = \langle\mathbf{N}\mathbf{N}\rangle$. The network is equivalent to maximum likelihood when this variance is equal to the Cramér-Rao bound [3], $\langle\delta\theta\rangle^2_{CR}$. If the noise, $\mathbf{N}$, is Gaussian with a covariance matrix independent of $\theta$, this bound is equal to:

$$\langle\delta\theta\rangle^2_{CR} = \frac{1}{\mathbf{F}' \cdot \mathbf{R}^{-1} \cdot \mathbf{F}'}. \tag{13}$$

For independent Gaussian noise of fixed variance, $\sigma^2$, and zero covariance, the variance of the network estimate, equation 12, becomes $\sigma^2/(|\mathbf{F}'|^2 \cos^2\mu)$ where $\mu$ is the angle between $\mathbf{v}^\dagger$ and $\mathbf{F}'$. The Cramér-Rao bound, on the other hand, is equal to $\sigma^2/|\mathbf{F}'|^2$. These expressions differ only by $\cos^2\mu$, which is 1 if $\mathbf{F} \propto \mathbf{v}^\dagger$. In addition, it is close to 1 for networks that have identical input and output tuning curves, $\mathbf{F}(\theta) = \mathbf{G}(\theta)$, *and* the Jacobian, $\mathbf{J}$, is nearly symmetric, so that $\mathbf{v} \approx \mathbf{v}^\dagger$ (recall that $\mathbf{v} = \mathbf{G}'$). If these last two conditions are satisfied, the network comes close to being a maximum likelihood estimator.

## 3.2   Application to Divisive Normalization

Divisive normalization is a particular example of the general case considered above. For simplicity, in our simulations we chose the input and output tuning curves to be equal ($\mathbf{F} = \mathbf{G}$ in the above notation), which lead to a value of 0.87 for $\cos^2\mu$ (evaluated numerically). This predicted a variance 15% above the Cramér-Rao

bound for independent Gaussian noise with fixed variance, consistent with the 16% we obtained in our simulations. The network also handles fairly well other noise distributions, such as Gaussian noise with variance proportional to the mean, as illustrated by our simulations.

## 4    Conclusions

We have recently shown that a subclass of line attractor networks can be used as maximum likelihood estimators[3]. This paper extend this conclusion to a much wider class of networks, namely, any network that admits a line (or, by straightforward extension of the above analysis, a higher dimensional) attractor. This is true in particular for networks using divisive normalization, a normalization which is thought to match quite closely the nonlinearity found in the primary visual cortex and MT.

Although our analysis relies on the existence of an attractor, this is not a requirement for obtaining near optimal noise filtering. As we have seen, 2-3 iterations are enough to achieve asymptotic performance (except at contrasts barely above threshold). What matters most is that our network implement a sequence of low pass filtering to filter out the noise, followed by a square nonlinearity to compensate for the widening of the tuning curve due to the low pass filter, and a normalization to weaken contrast dependence. It is likely that this process would still clean up noise efficiently in the first 2-3 iterations even if activity decayed to zero eventually, that is to say, even if the hills of activity were not stable states. This would allow us to apply our approach to other types of networks, including those lacking circular symmetry and networks with continuously clamped inputs.

To conclude, we propose that each cortical layer may read out the activity in the preceding layer in an optimal way thanks to the nonlinear pooling properties of divisive normalization, and, as a result, may behave like an ideal observer. It is therefore possible that the ability to read out neuronal codes in the sensory cortices in an optimal way may not be confined to a few areas like the parietal or frontal cortex, but may instead be a general property of every cortical layer.

## Footnotes

[1] The line attractor is, in fact, an idealization; for $P$ units the attractors associated with equation 5 consists of $P$ isolated points. However, for $P$ large, the attractors are spaced closely enough that they may be considered a line.

## References

[1] D. Heeger. Normalization of cell responses in cat striate cortex. *Visual Neuroscience*, 9:181–197, 1992.

[2] L. Itti, C. Koch, and J. Braun. A quantitative model for human spatial vision threshold on the basis of non-linear interactions among spatial filters. In R. Lippman, J. Moody, and D. Touretzky, editors, *Advances in Neural Information Processing Systems*, volume 11. Morgan-Kaufmann, San Mateo, 1998.

[3] A. Pouget, K. Zhang, S. Deneve, and P. Latham. Statistically efficient estimation using population coding. *Neural Computation*, 10:373–401, 1998.

[4] E. Simoncelli and D. Heeger. A model of neuronal responses in visual area MT. *Vision Research*, 38(5):743–761, 1998.